# Fast biped walking with a reflexive controller and real-time policy searching

**Tao Geng**[1], **Bernd Porr**[2] **and Florentin Wörgötter**[1,3]

[1] Dept. Psychology, University of Stirling, UK.

`runbot05@gmail.com`

[2] Dept. Electronics & Electrical Eng., University of Glasgow, UK.

`b.porr@elec.gla.ac.uk`

[3] Bernstein Centre for Computational Neuroscience, University of Göttingen

`worgott@chaos.gwdg.de`

## Abstract

In this paper, we present our design and experiments of a planar biped robot ("RunBot") under pure reflexive neuronal control. The goal of this study is to combine neuronal mechanisms with biomechanics to obtain very fast speed and the on-line learning of circuit parameters. Our controller is built with biologically inspired sensor- and motor-neuron models, including local reflexes and not employing any kind of position or trajectory-tracking control algorithm. Instead, this reflexive controller allows RunBot to exploit its own natural dynamics during critical stages of its walking gait cycle. To our knowledge, this is the first time that dynamic biped walking is achieved using only a pure reflexive controller. In addition, this structure allows using a policy gradient reinforcement learning algorithm to tune the parameters of the reflexive controller in real-time during walking. This way RunBot can reach a relative speed of 3.5 leg-lengths per second after a few minutes of online learning, which is faster than that of any other biped robot, and is also comparable to the fastest relative speed of human walking. In addition, the stability domain of stable walking is quite large supporting this design strategy.

## 1  Introduction

Building and controlling fast biped robots demands a deeper understanding of biped walking than for slow robots. While slow robots may walk statically, fast biped walking has to be dynamically balanced and more robust as less time is available to recover from disturbances [1]. Although many biped robots have been developed using various technologies in the past 20 years, their walking speeds are still not comparable to that of their counterpart in nature, humans. Most of the successful biped robots have commonly used the ZMP (Zero Moment Point, [2]) as the criterion for stability control and motion generation. The ZMP is the point on the ground where the total moment generated by gravity and inertia equals zero. This measure has two deficiencies in the case of high-speed walking. First, the ZMP must always reside in the convex hull of the stance foot, and the stability margin is measured by the minimal distance between the ZMP and the edge of the foot. To ensure

an appropriate stability margin, the foot has to be flat and large, which will deteriorate the robot's performance and pose great difficulty during fast walking. This difficulty can be shown clearly when humans try to walk with skies or swimming fins. Second, the ZMP criterion does not permit rotation of the stance foot at the heel or the toe, which, however, can amount to up to eighty percent of a normal human walking gait, and is important and inevitable in fast biped walking.

On the other hand, sometimes dynamic biped walking can be achieved without considering any stability criterion such as the ZMP. For example, passive biped robots can walk down a shallow slope without sensing or control. Some researchers have proposed approaches to equip a passive biped with actuators to improve its performance and drive it to walk on the flat ground [3] [4]. Nevertheless, these passive bipeds excessively depend on their natural dynamics for gait generation, which, while making their gaits efficient in energy, also limits their walking rate to be very slow.

In this study, we will show that, with a properly designed mechanical structure, a novel, pure reflexive controller, and an online policy gradient reinforcement learning algorithm, our biped robot can attain a fast walking speed of 3.5 leg-lengths per second. This makes it faster than any other biped robot we know. Though not a passive biped, it exploits its own natural dynamics during some stages of its walking gait, greatly simplifying the necessary control structures.

## 2 The robot

RunBot (Fig. 1) is 23 cm high, foot to hip joint axis. It has four joints: left hip, right hip, left knee, right knee. Each joint is driven by a modified RC servo motor. A hard mechanical stop is installed on the knee joints, preventing it from going into hyperextension. Each foot is equipped with a modified piezo transducer to sense ground contact events. Similar to other approaches [1], we constrain the robot only in the sagittal plane by a boom of one meter length freely rotating in its joints (planar robot). This assures that RunBot can still very easily trip and fall in the sagittal plane.

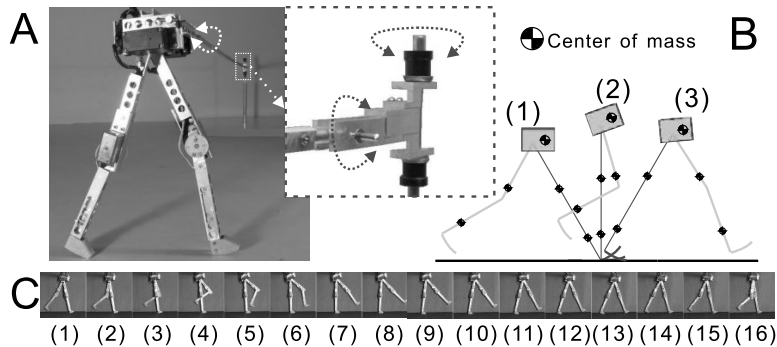

Figure 1: A): The robot, RunBot, and its boom structure. All three orthogonal axis of the boom can rotate freely. B) Illustration of a walking step of RunBot. C) A series of sequential frames of a walking gait cycle. The interval between every two adjacent frames is 33 ms. Note that, during the time between frame (8) and frame (13), which is nearly one third of the duration of a step, the motor voltage of all four joints remain to be zero, and the whole robot is moving passively. At the time of frame (13), the swing leg touches the floor and a next step begins.

Since we intended to exploit RunBot's natural dynamics during some stages of its gait

cycle; similar to passive bipeds; its foot bottom is also curved with a radius equal to half the leg-length (with a too large radius, the tip of the foot may strike the ground during its swing phase). During the stance phase of such a curved foot, always only one point touches the ground, thus allowing the robot to roll passively around the contact point, which is similar to the rolling action of human feet facilitating fast walking.

The most important consideration in the mechanical design of our robot is the location of its center of mass. About seventy percent of the robot's weight is concentrated on its trunk. The parts of the trunk are assembled in such a way that its center of mass is located before the hip axis (Fig. 1 A). The effect of this design is illustrated in Fig. 1 B. As shown, one walking step includes two stages, the first from (1) to (2), the second from (2) to (3). During the first stage, the robot has to use its own momentum to rise up on the stance leg. When walking at a low speed, the robot may have not enough momentum to do this. So, the distance the center of mass has to cover in this stage should be as short as possible, which can be fulfilled by locating the center of mass of the trunk forward. In the second stage, the robot just falls forward naturally and catches itself on the next stance leg. Then the walking cycle is repeated. The figure also shows clearly the rolling movement of the curved foot of the stance leg. A stance phase begins with the heel touching ground, and terminates with the toe leaving ground.

In summary, our mechanical design of RunBot has following special features that distinguish it from other powered biped robots and facilitate high-speed walking and exploitation of natural dynamics: (a) Small curved feet allowing for rolling action; (b) Unactuated, hence, light ankles; (c) Light-weight structure; (d) Light and fast motors; (e) Proper mass distribution of the limbs; (f) Properly positioned mass center of the trunk.

## 3 The neural structure of our reflexive controller

The reflexive walking controller of RunBot follows a hierarchical structure (Fig. 2). The bottom level is the reflex circuit local to the joints, including motor-neurons and angle sensor neurons involved in the joint reflexes. The top level is a distributed neural network consisting of hip stretch receptors and ground contact sensor neurons, which modulate the local reflexes of the bottom level. Neurons are modelled as non-spiking neurons simulated on a Linux PC, and communicated to the robot via the DA/AD board. Though somewhat simplified, they still retain some of the prominent neuronal characteristics.

### 3.1 Model neuron circuit of the top level

The joint coordination mechanism in the top level is implemented with the neuron circuit illustrated in Fig. 2. While other biologically inspired locomotive models and robots use two stretch receptors on each leg to signal the attaining of the leg's AEP (Anterior Extreme Position) and PEP (Posterior Extreme Position) respectively, our robot has only one stretch receptor on each leg to signal the AEA (Anterior Extreme Angle) of its hip joint. Furthermore, the function of the stretch receptor on our robot is only to trigger the extensor reflex on the knee joint of the same leg, rather than to implicitly reset the phase relations between different legs as in the case of Cruse's model. As the hip joint approaches the AEA, the output of the stretch receptors for the left (AL) and the right hip (AR) are increased as:

$$\rho_{AL} = \left(1 + e^{\alpha_{AL}(\Theta_{AL} - \phi)}\right)^{-1} \tag{1}$$

$$\rho_{AL} = \left(1 + e^{\alpha_{AR}(\Theta_{AR} - \phi)}\right)^{-1} \tag{2}$$

Where $\phi$ is the real time angular position of the hip joint, $\Theta_{AL}$ and $\Theta_{AR}$ are the hip anterior extreme angles whose values are tuned by hand, $\alpha_{AL}$ and $\alpha_{AR}$ are positive constants. This

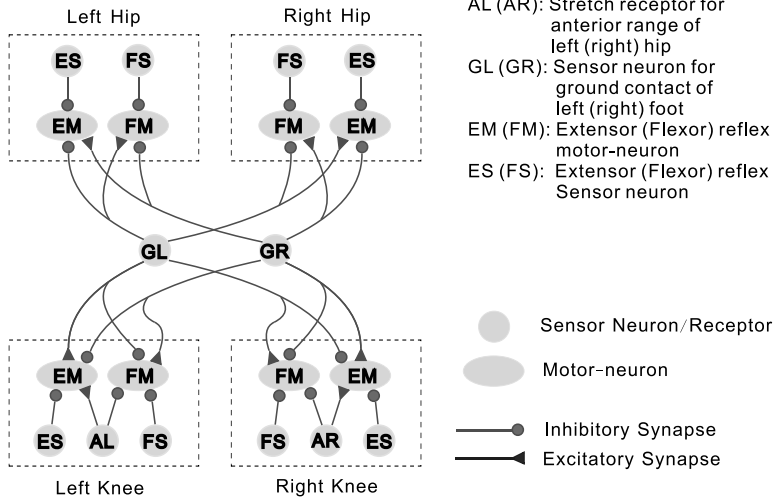

Figure 2: The neuron model of reflexive controller on RunBot.

model is inspired by a sensor neuron model presented in [5] that is thought capable of emulating the response characteristics of populations of sensor neurons in animals. Another kind of sensor neuron incorporated in the top level is the ground contact sensor neuron, which is active when the foot is in contact with the ground. Its output, similar to that of the stretch receptors, changes according to:

$$\rho_{GL} \quad = \quad \left(1 + e^{\alpha_{GL}(\Theta_{GL} - V_L + V_R)}\right)^{-1} \tag{3}$$

$$\rho_{GR} \quad = \quad \left(1 + e^{\alpha_{GR}(\Theta_{GR} - V_R + V_L)}\right)^{-1} \tag{4}$$

Where $V_L$ and $V_R$ are the output voltage signals from piezo sensors of the left foot and right foot respectively, $\Theta_{GL}$ and $\Theta_{GR}$ work as thresholds, $\alpha_{GL}$ and $\alpha_{GR}$ are positive constants.

### 3.2 Neural circuit of the bottom level

The bottom-level reflex system of our robot consists of reflexes local to each joint (Fig. 2). The neuron module for one reflex is composed of one angle sensor neuron and the motor-neuron it contacts. Each joint is equipped with two reflexes, extensor reflex and flexor reflex, both are modelled as a monosynaptic reflex, that is, whenever its threshold is exceeded, the angle sensor neuron directly excites the corresponding motor-neuron. This direct connection between angle sensor neuron and motor-neuron is inspired by a reflex described in cockroach locomotion [6]. In addition, the motor-neurons of the local reflexes also receive an excitatory synapse and an inhibitory synapse from the neurons of the top level, by which the top level can modulate the bottom-level reflexes. Each joint has two angle sensor neurons, one for the extensor reflex, and the other for the flexor reflex (Fig. 2). Their models are similar to that of the stretch receptors described above. The extensor angle sensor neuron changes its output according to:

$$\rho_{ES} = \left(1 + e^{\alpha_{ES}(\Theta_{ES} - \phi)}\right)^{-1} \tag{5}$$

where $\phi$ is the real time angular position obtained from the potentiometer of the joint. $\Theta_{ES}$ is the threshold of the extensor reflex and $\alpha_{ES}$ a positive constant. Likewise, the output of

Table 1: Parameters of neurons for hip- and knee joints. For meaning of the subscripts, see Fig. 2.

|  | $\Theta_{EM}$ | $\Theta_{FM}$ | $\alpha_{ES}$ | $\alpha_{FS}$ |
|---|---|---|---|---|
| Hip Joints | 5 | 5 | 2 | 2 |
| Knee Joints | 5 | 5 | 2 | 2 |

Table 2: Parameters of stretch receptors and ground contact sensor neurons.

| $\Theta_{GL}$ (v) | $\Theta_{GR}$ (v) | $\Theta_{AL}$ (deg) | $\Theta_{AR}$ (deg) | $\alpha_{GL}$ | $\alpha_{GR}$ | $\alpha_{AL}$ | $\alpha_{AR}$ |
|---|---|---|---|---|---|---|---|
| 2 | 2 | $= \Theta_{ES}$ | $= \Theta_{ES}$ | 2 | 2 | 2 | 2 |

the flexor sensor neuron is modelled as:

$$\rho_{FS} = (1 + e^{\alpha_{FS}(\phi - \Theta_{FS})})^{-1} \qquad (6)$$

with $\Theta_{FS}$ and $\alpha_{FS}$ similar as above. The direction of extensor on both hip and knee joints is forward while that of flexors is backward.

It should be particularly noted that the thresholds of the sensor neurons in the reflex modules do not work as desired positions for joint control, because our reflexive controller does not involve any exact position control algorithms that would ensure that the joint positions converge to a desired value. The motor-neuron model is adapted from one used in the neural controller of a hexapod simulating insect locomotion [7]. The state and output of each extensor motor-neuron is governed by equations 7,8 [8] (that of flexor motor-neurons are similar):

$$\tau \frac{dy}{dt} = -y + \sum \omega_X \rho_X \qquad (7)$$

$$u_{EM} = \left(1 + e^{\Theta_{EM} - y}\right)^{-1} \qquad (8)$$

Where $y$ represents the mean membrane potential of the neuron. Equation 8 is a sigmoidal function that can be interpreted as the neuron's short-term average firing frequency, $\Theta_{EM}$ is a bias constant that controls the firing threshold. $\tau$ is a time constant associated with the passive properties of the cell membrane [8], $\omega_X$ represents the connection strength from the sensor neurons and stretch receptors to the motor-neuron neuron (Fig. 2). $\rho_X$ represents the output of the sensor-neurons and stretch receptors that contact this motor-neuron (e.g., $\rho_{ES}, \rho_{AL}, \rho_{GL}$, etc.)

Note that, on RunBot, the output value of the motor-neurons, after multiplication by a gain coefficient, is sent to the servo amplifier to directly drive the joint motor. The voltage of joint motor is determined by

$$Motor\ Voltage = M_{AMP} G_M (s_{EM} u_{EM} + s_{FM} u_{FM}), \qquad (9)$$

where $M_{AMP}$ represents the magnitude of the servo amplifier, which is 3 on RunBot. $G_M$ stands for output gain of the motor-neurons. $s_{EM}$ and $s_{FM}$ are signs for the motor voltage of flexor and extensor, being +1 or -1, depending on the the hardware of the robot. $u_{EM}$ and $u_{FM}$ are the outputs of the motor-neurons.

## 4 Robot walking experiments

The model neuron parameters chosen jointly for all experiments are listed in Tables 1 and 2. The time constants $\tau_i$ of all neurons take the same value of 3ms. The weights of all

Table 3: Fixed parameters of the knee joints.

| | $\Theta_{ES,k}$ (deg) | $\Theta_{FS,k}$ (deg) | $G_{M,k}$ |
|---|---|---|---|
| Knee Joints | 175 | 110 | $0.9G_{M,h}$ |

the inhibitory connections are set to -10, except those between sensor-neurons and motor-neurons, which are -30, and those between stretch receptors and flexor motor-neurons, which are -15. The weights of all excitatory connections are 10, except those between stretch receptors and extensor motor-neurons, which are 15. Because the movements of the knee joints is needed mainly for timely ground clearance without big contributions to the walking speed, we set their neuron parameters to fixed values (see Table 3 ). We also fix the threshold of the flexor sensor neurons of the hips ($\Theta_{FS,h}$) to $85°$. So, in the experiments described below, we only need to tune the two parameters of the hip joints, the threshold of the extensor sensor neurons ($\Theta_{ES,h}$) and the gain of the motor-neurons ($G_{M,h}$), which work together to determine the walking speed and the important gait properties of RunBot. In RunBot, $\Theta_{ES,h}$ determines roughly the stride length (not exactly, because the hip joint moves passively after passing $\Theta_{ES,h}$), while $G_{M,h}$ is proportional to the angular velocity of the motor on the hip joint.

In experiments of walking on a flat floor, surprisingly, we have found that stable gaits can appear in a considerably large range of the parameters $\Theta_{ES,h}$ and $G_{M,h}$ (Fig. 3A).

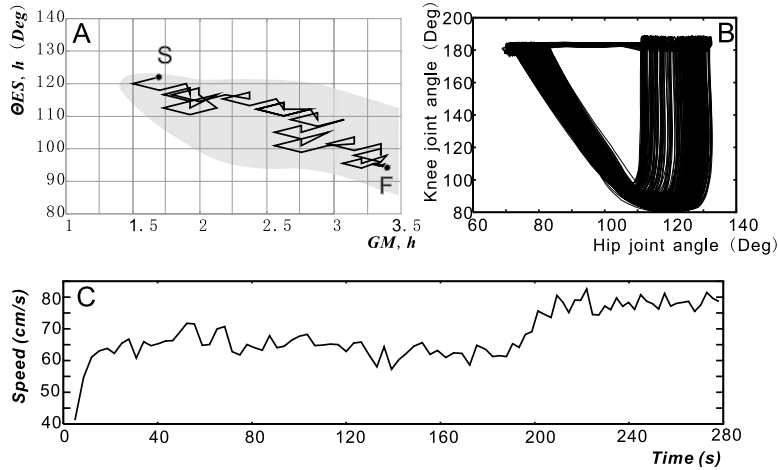

Figure 3: (A), The range of the two parameters, $G_{M,h}$ and $\Theta_{ES,h}$, in which stable gaits appear. The maximum permitted value of $G_{M,h}$ is 3.5 (higher value will destroy the motor of the hip joint). See text for more information. (B), Phase diagrams of hip joint position and knee joint position of one leg during the whole learning process. The smallest orbit is the fastest walking gait. (C), The walking speed of RunBot during the learning process.

In RunBot, passive movements appear on two levels, at the single joint level and at the whole robot level. Due to the high gear ratio of the joint motors, the passive movement of each joint is not very large. Whereas the effects of passive movements at the whole robot level can be clearly seen especially when RunBot is walking at a medium or slow speed (Fig. 1 C).

### 4.1 Policy gradient searching for fast walking gaits

In order to get a fast walking speed, the biped robot should have a long stride length, a short swing time, and a short double support phase [1]. In RunBot, because the phase-switching of its legs is triggered immediately by ground contact signals, its double support phase is so short (usually less than 30 ms) that it is negligible. A long stride length and a short swing time are mutually exclusive. Because there are no position or trajectory tracking control in RunBot, it is impossible to control its walking speed directly or explicitly. However, knowing that runBot's walking gait is determined by only two parameters, $\Theta_{ES,h}$ and $G_{M,h}$ (Fig. 3A), we formulate RunBot's fast walking control as a policy gradient reinforcement learning problem by considering each point in the the parameter space (Fig. 3A) as an open-loop policy that can be executed by RunBot in real-time.

Our approach is modified from [9]. It starts from an initial parameter vector $\pi = (\theta_1, \theta_2)$ (here $\theta_1$ and $\theta_2$ represent $G_{M,h}$ and $\Theta_{ES,h}$, respectively) and proceeds to evaluate following 5 polices near $\pi$: $(\theta_1, \theta_2), (\theta_1, \theta_2 + \epsilon_2), (\theta_1 - \epsilon_1, \theta_2), (\theta_1, \theta_2 - \epsilon_2), (\theta_1 + \epsilon_1, \theta_2)$, where each $\epsilon_j$ is a adaptive value that is small relative to $\theta_j$. The evaluation of each policy generates a score that is a measure of the speed of the gait described by that policy. We use these scores to construct an adjustment vector $A$ [9]. Then $A$ is normalized and multiplied by an adaptive step-size. Finally, we add $A$ to $\pi$, and begin the next iteration. If $A = 0$, this means a possible local minimum is encountered. In this case, we replace $A$ with a stochastically generated vector. Although this is a very simple strategy, our experiments show that it can effectively prevent the real-time learning from trapping in the local minimums.

One experiment result is shown in Fig. 3. RunBot starts its walking with the parameters corresponding to point S in Fig. 3A whose speed is 41 cm/s (see Fig. 3C). After 240 seconds of continuous walking with the learning algorithm and no any human intervention, RunBot attains a walking speed of about 80 cm/s (see Fig. 3C, corresponding to point F in Fig. 3A), which is equivalent to 3.5 leg-lengths per second. To compare the walking speed of various biped robots whose sizes are quite different from each other, we use the relative speed, speed divided by the leg-length. We know of no other biped robot attaining such a fast relative speed. The world record of human walking race is equivalent to about $4.0 - 4.5$ leg-lengths per second. So, RunBot's highest walking speed is comparable to that of humans. To get a feeling of how fast RunBot can walk, we strongly encourage readers to watch the videos of the experiment at, http://www.cn.stir.ac.uk/~tgeng/nips

Although there is no specifically designed controller in charge of the sensing and control of the transient stages of policy changing (speed changing), the natural dynamics of the robot itself ensures the stability during the changes. By exploiting the natural dynamics, the reflexive controller is robust to its parameter variation as shown in Fig. 3A.

## 5 Discussions

Cruse developed a completely decentralized reflexive controller model to understand the locomotion control of walking in stick insects (Carausius morosus, [10]), which can immensely decrease the computational burden of the locomotion controller, and has been applied in many hexapod robots. Up to date, however, no real biped robot has existed that depends exclusively on reflexive controllers. This may be because of the intrinsic instability specific to biped-walking, which makes the dynamic stability of biped robots much more difficult to control than that of multi-legged robots. To our knowledge, our RunBot is the first dynamic biped exclusively controlled by a pure reflexive controller. Although such a pure reflexive controller itself involves no explicit mechanisms for the global stability control of the biped, its coupling with the properly designed mechanics of RunBot has substantially ensured the considerably large stable domain of the dynamic biped gaits.

Our reflexive controller has some evident differences from Cruse's model. Cruse's model depends on PEP, AEP and GC (Ground Contact) signals to generate the movement pattern of the individual legs. Whereas our reflexive controller presented here uses only GC and AEA signals to coordinate the movements of the joints. Moreover, the AEA signal of one hip in RunBot only acts on the knee joint belonging to the same leg, not functioning on the leg-level as the AEP and PEP did in Cruse's model. The use of fewer phasic feedback signals has further simplified the controller structure in RunBot.

In order to achieve real time walking gait in a real world, even biological inspired robots often have to depend on some kinds of position- or trajectory tracking control on their joints [6, 11, 12]. However, in RunBot, there is no exact position control implemented. The neural structure of our reflexive controller does not depend on, or ensure the tracking of, any desired position. Indeed, it is this approximate nature of our reflexive controller that allows the physical properties of the robot itself to contribute implicitly to generation of overall gait trajectories. The effectiveness of this hybrid neuro-mechanical system is also reflected in the fact that real-time learning of parameters was possible, where sometimes the speed of the robot changes quite strongly (see movie) without tripping it.

## References

[1] J. Pratt. *Exploiting Inherent Robustness and Natural Dynamics in the Control of Bipedal Walking Robots*. PhD thesis, Massachusetts Institute of Technology, 2000.

[2] B. Surla D. Vukobratovic, M. Borovac and D. Stokic. *Biped locomotion: dynamics, stability, control and application*. Springer-Verlag, 1990.

[3] R. Q. V. Van der Linde. Active leg compliance for passive walking. In *Proceedings of IEEE International Conference on Robotics and Automation*, Orlando, Florida, 1998.

[4] Steve Collins and Andy Ruina. Efficient bipedal robots based on passive-dynamic walkers. *Science*, 37:1082–1085, 2005.

[5] T. Wadden and O. Ekeberg. A neuro-mechanical model of legged locomotion: Single leg control. *Biological Cybernetics*, 79:161–173, 1998.

[6] R.D. Beer, R.D. Quinn, H.J. Chiel, and R.E. Ritzmann. Biologically inspired approaches to robotics. *Communications of the ACM*, 40(3):30–38, 1997.

[7] R.D. Beer and H.J. Chiel. A distributed neural network for hexapod robot locomotion. *Neural Computation*, 4:356–365, 1992.

[8] J.C. Gallagher, R.D. Beer, K.S. Espenschied, and R.D. Quinn. Application of evolved locomotion controllers to a hexapod robot. *Robotics and Autonomous Systems*, 19:95–103, 1996.

[9] Nate Kohl and Peter Stone. Policy gradient reinforcement learning for fast quadrupedal locomotion. In *Proceedings of the IEEE International Conference on Robotics and Automation*, volume 3, pages 2619–2624, May 2004.

[10] H. Cruse, T. Kindermann, M. Schumm, and et.al. Walknet - a biologically inspired network to control six-legged walking. *Neural Networks*, 11(7-8):1435–1447, 1998.

[11] Y. Fukuoka, H. Kimura, and A.H. Cohen. Adaptive dynamic walking of a quadruped robot on irregular terrain based on biological concepts. *Int. J. of Robotics Research*, 22:187–202, 2003.

[12] M.A. Lewis. Certain principles of biomorphic robots. *Autonomous Robots*, 11:221–226, 2001.
